# ANALOG IMPLEMENTATION OF SHUNTING NEURAL NETWORKS

Bahram Nabet, Robert B. Darling, and Robert B. Pinter
Department of Electrical Engineering, FT-10
University of Washington
Seattle, WA 98195

## ABSTRACT

An extremely compact, all analog and fully parallel implementation of a class of shunting recurrent neural networks that is applicable to a wide variety of FET-based integration technologies is proposed. While the contrast enhancement, data compression, and adaptation to mean input intensity capabilities of the network are well suited for processing of sensory information or feature extraction for a content addressable memory (CAM) system, the network also admits a global Liapunov function and can thus achieve stable CAM storage itself. In addition the model can readily function as a front-end processor to an analog adaptive resonance circuit.

## INTRODUCTION

Shunting neural networks are networks in which multiplicative, or shunting, terms of the form $x_i \sum_j f_j(x_j)$ or $x_i \sum_j I_j$ appear in the short term memory equations, where $x_i$ is activity of a cell or a cell population or an iso-potential portion of a cell and $I_i$ are external inputs arriving at each site. The first case shows recurrent activity, while the second case is non-recurrent or feed forward. The polarity of these terms signify excitatory or inhibitory interactions.

Shunting network equations can be derived from various sources such as the passive membrane equation with synaptic interaction (Grossberg 1973, Pinter 1983), models of dendritic interaction (Rall 1977), or experiments on motoneurons (Ellias and Grossberg 1975).

While the exact mechanisms of synaptic interactions are not known in every individual case, neurobiological evidence of shunting interactions appear in several

areas such as sensory systems, cerebellum, neocortex, and hippocampus (Grossberg 1973, Pinter 1987). In addition to neurobiology, these networks have been used to successfully explain data from disciplines ranging from population biology (Lotka 1956) to psychophysics and behavioral psychology (Grossberg 1983).

Shunting nets have important advantages over additive models which lack the extra nonlinearity introduced by the multiplicative terms. For example, the total activity of the network, shown by $\sum_i x_i$, approaches a constant even as the input strength grows without bound. This normalization in addition to being computationally desirable has interesting ramifications in visual psychophysics (Grossberg 1983). Introduction of multiplicative terms also provides a negative feedback loop which automatically controls the gain of each cell, contributes to the stability of the network, and allows for large dynamic range of the input to be processed by the network. The automatic gain control property in conjunction with properly chosen nonlinearities in the feedback loop makes the network sensitive to small input values by suppressing noise while not saturating at high input values (Grossberg 1973). Finally, shunting nets have been shown to account for short term adaptation to input properties, such as adaptation level tuning and the shift of sensitivity with background strength (Grossberg 1983), dependence of visual size preference and latency of response on contrast and mean luminance, and dependence of temporal and spatial frequency tuning on contrast and mean luminance (Pinter 1985).

## IMPLEMENTATION

The advantages, generality, and applicability of shunting nets as cited in the previous section make their implementation very desirable, but digital implementation of these networks is very inefficient due to the need for analog to digital conversion, multiplication and addition instructions, and implementation of iterative algorithms. A linear feedback class of these networks $(x_i \sum_j f_j(x_j) = x_i \sum_j K_{ij} x_j)$, however, can be implemented very efficiently with simple, completely parallel and all analog circuits.

## FRAMEWORK

Figure 1 shows the design framework for analog implementation of a class of shunting nets. In this design addition (subtraction) is achieved, via Kirchoff's current law by placing transistors in upper (lower) rails, and through the choice of depletion or enhancement mode devices. Multiplicative, or shunting, interconnections are done by *one* transistor per interconnect, using a field-effect transistor (FET) in the voltage-variable conductance region. Temporal properties are characterized by cell membrane capacitance C, which can be removed, or in effect replaced by the parasitic device capacitances, if higher speed is desired. A buffer stage is necessary for correct polarity of interconnections and the large fan-out associated with high connectivity of neural networks.

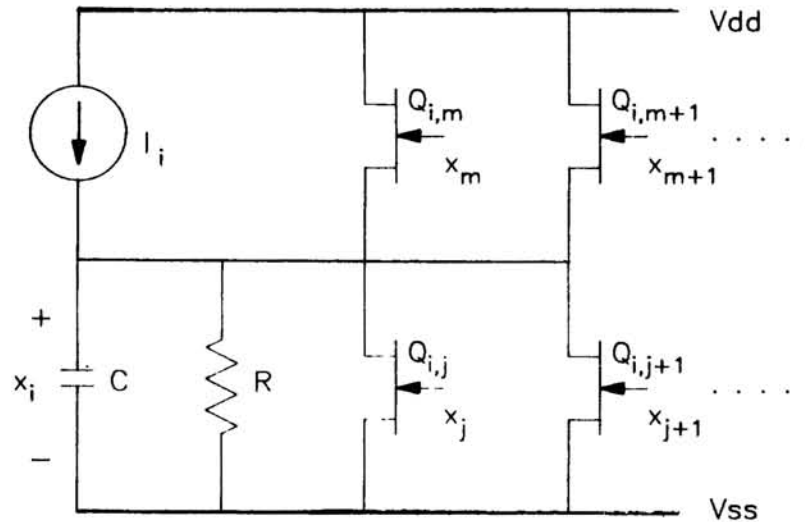

**Figure 1.** *Design framework for implementation of one cell in a shunting network. Voltage output of other cells is connected to the gate of transistors $Q_{i,j}$.*

Such a circuit is capable of implementing the general network equation:

$$\frac{dx_i}{dt} = \pm I_i \pm a_i x_i \pm x_i(K_i x_i) \pm x_i \sum_{j \neq i} K_{ij} x_j \pm \sum_j K_j x_j^2 \qquad (1)$$

Excitatory and inhibitory input current sources can also be shunted, with extra circuitry, to implement non-recurrent shunting networks.

## NMOS, CMOS and GALLIUM ARSENIDE

Since the basic cell of Fig. 1 is very similar to a standard logic gate inverter, but with the transistors sized by gate width-to-length ratio to operate in the nonsaturated current region, this design is applicable to a variety of FET technologies including NMOS, CMOS, and gallium arsenide (GaAs).

A circuit made of all depletion-mode devices such as GaAs MESFET buffered FET logic, can implement all the terms of Eq. (1) except shunting excitatory terms and requires a level shifter in the buffer stage. A design with all enhancement mode devices such as silicon NMOS can do the same but without a level shifter. With the addition of p-channel devices, e.g. Si CMOS, all polarities and all terms of Eq. (1) can be realized. As mentioned previously a buffer stage is necessary for correct polarity of interconnections and fan out/fan in capacity.

Figure 2 shows a GaAs MESFET implementation with only depletion mode devices which employs a level shifter as the buffer stage.

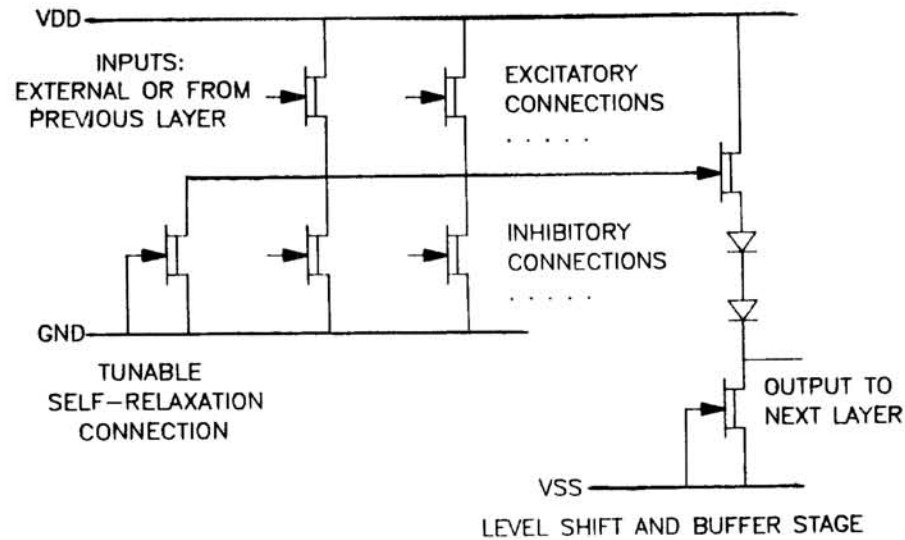

**Figure 2.** *Gallium arsenide MESFET implementation with level shifter and depletion mode devices. Lower rail transistors produce shunting off-surround terms. Upper transistors can produce additive excitatory connections.*

## SPECIFIC IMPLEMENTATION

The simplest shunting network that can be implemented by the general framework of Fig.1 is Fig. 2 with only inhibitory connections (lower rail transistors). This circuit implements the network model

$$\frac{dX_i}{dt} = I_i - a_i X_i + X_i(K_i X_i) - X_i(\sum_{j \neq i} K_{ij} X_j) \tag{2}$$

The simplicity of the implementation is notable; a linear array *with* nearest neighbor interconnects consists of only 5 transistors, 1–3 diodes, and if required 1 capacitor per cell.

A discrete element version of this implementation has been constructed and shows good agreement with expected properties. Steady state output is proportional to the square root of a uniform input thereby *compressing* the input data and showing *adaptation to mean input intensity* (figure 3). The network exhibits *contrast enhancement* of spatial edges which increases with higher mean input strength (figure 4). A point source input elicits an on-center off-surround response, similar to the difference-of-Gaussians receptive field of many excitable cells. This *'receptive field'* becomes more pronounced as the input intensity increases, showing the *dependence of spatial frequency tuning on mean input level* (figure 5). The temporal response of the network is also input dependent since the time constant of the exponential

decay of the impulse response decreases with input intensity. Finally, the dependence of the above properties on mean input strength can be tuned by varying the conductance of the central FET.

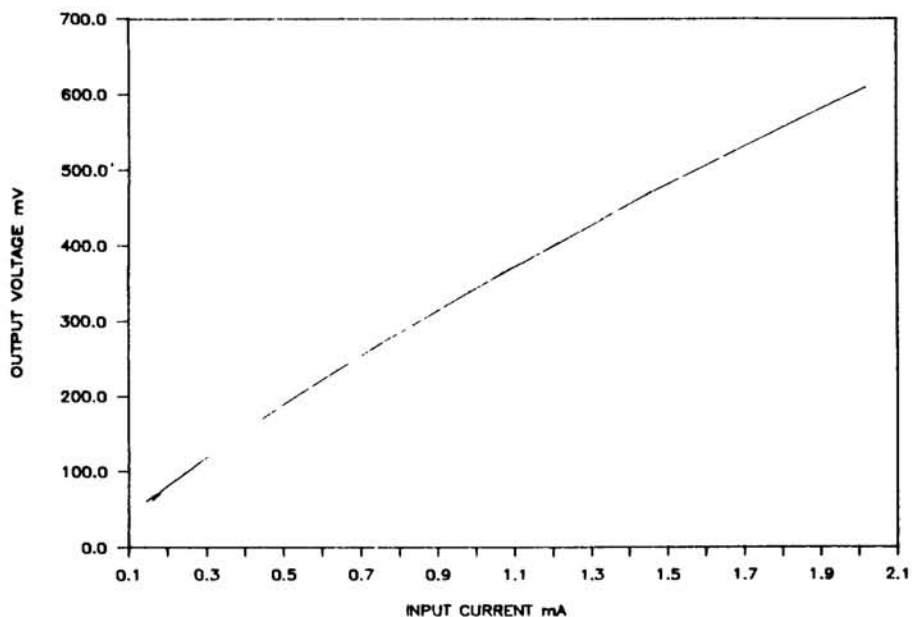

**Figure 3.** *Response of network to uniform input. Output is proportional to the square root of the input.*

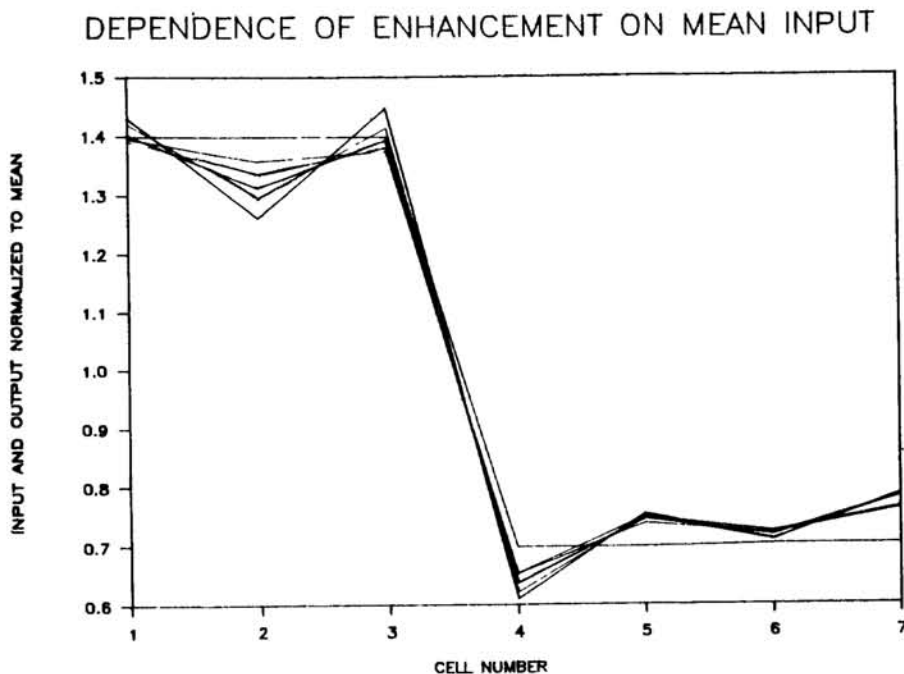

**Figure 4.** *Response of network to spatial edge patterns with the same contrast but increasing mean input level.*

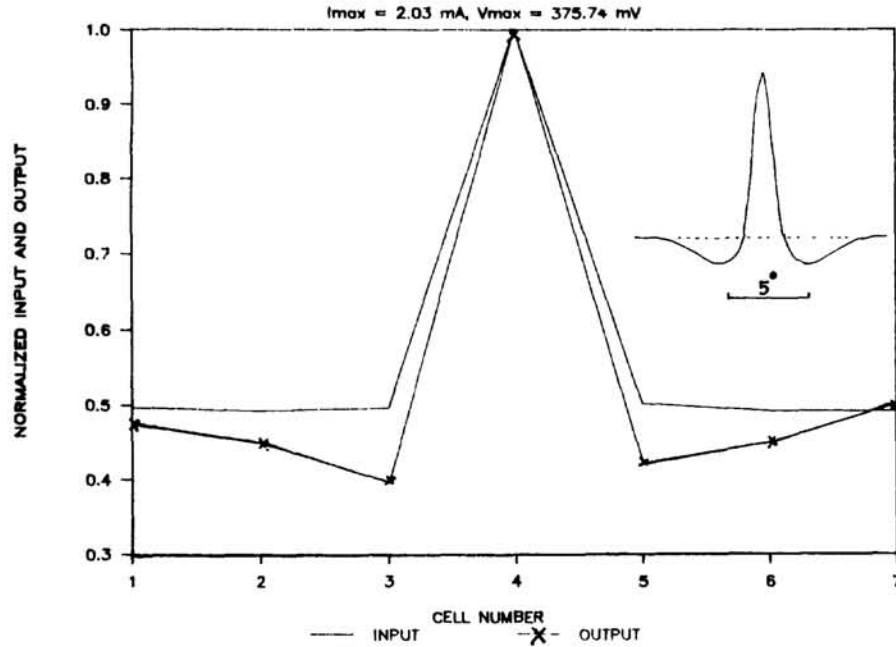

**Figure 5.** *Response of network to a point source input. Inset shows the receptive field of fly's lamina monopolar cells (LMC of Lucilia sericata). Horizontal axis of inset in visual angle, vertical axis relative voltage units of hyperpolarization. Inset from Pinter et al. (in preparation)*

## CONTENT ADDRESSABILITY AND RELATION TO ART

Using a theorem by Cohen and Grossberg (1983), it can be shown that the network equation (2) admits the global Liapunov function

$$v = -\sum_{i=1}^{n}\left(I_i \ln(\frac{x_i}{\lambda}) - a_i x_i + K_i x_i^2\right) + \frac{1}{2}\sum_{j,k=1}^{n} K_{ij} x_j x_k, \qquad (3)$$

where $\lambda$ is a constant, under the constraints $K_{ij} = K_{ji}$ and $x_i > 0$. This shows that in response to an arbitrary input the network always approaches an equilibrium point. The equilibria represent stored patterns and this is Content Addressable Memory (CAM) property.

In addition, Eq. (2) is a special case of the feature representation field of an analog adaptive resonance theory ART-2 circuit, (Carpenter and Grossberg 1987), and hence this design can operate as a module in a learning multilayer ART architecture.

# FUTURE PLANS

Due to the very small number of circuit components required to construct a cell, this implementation is quite adaptable to very high integration densities. A solid state implementation of the circuit of figure (2) on a gallium arsenide substrate, chosen for its superiority for opto-electronics applications, is in progress. The chip includes monolithically fabricated photosensors for processing of visual information. All of the basic components of the circuit have been fabricated and tested. With standard 2 micron GaAs BFL design rules, a chip could contain over 1000 cells per $cm^2$, assuming an average of 20 inputs per cell.

# CONCLUSIONS

The present work has the following distinguishing features:
- Implements a mathematically well described and stable model.
- Proposes a *framework* for implementation of shunting nets which are biologically feasible, explain variety of psychophysical and psychological data and have many desirable computational properties.
- Has self-sufficient computational capabilities; especially suited for processing of sensory information in general and visual information in particular (Nabet and Darling 1988).
- Produces a 'good representation' of the input data which is also compatible with the self-organizing multilayer neural network architecture ART-2.
- Is suitable for implementation in variety of technologies.
- Is parallel, analog, and has very little overhead circuitry.

# REFERENCES

Carpenter, G.A. and Grossberg, S. (1987) "ART 2: self organization of stable category recognition codes for analog input patterns,". *Applied Optics* 26, pp. 4919–4930.

Cohen,M.A. and Grossberg, S. (1983) "Absolute stability of global pattern formation and parallel memory storage by competitive neural networks", *IEEE Transactions on Systems Man and Cybernetics* SMC-13, pp. 815–826.

Ellias, S.A. and Grossberg, S. (1975) "Pattern formation, contrast control, and oscillations in the short term memory of shunting on-center off-surround networks" *Biological Cybernetics,* 20, pp. 69–98.

Grossberg, S. (1973), "Contour enhancement, Short term memory and constancies in reverberating neural networks," *Studies in Applied Mathematics,* 52, pp. 217–257.

Grossberg, S. (1983), "The quantized geometry of visual space: the coherent computation of depth, form, and lightness." *The behavioral and brain sciences,* 6, pp. 625–692.

Lotka, A.J. (1956). *Elements of mathematical biology.* New York: Dover.

Nabet, B. and Darling, R.B. (1988). "Implementation of optical sensory neural networks with simple discrete and monolithic circuits," (Abstract) *Neural Networks,* Vol.1, Suppl. 1, 1988, pp. 396.

Pinter, R.B., (1983). "The electrophysiological bases for linear and nonlinear product term lateral inhibition and the consequences for wide-field textured stimuli" *J.Theor. Biol.* 105 pp. 233–243.

Pinter, R.B. (1985) " Adaptation of spatial modulation transfer functions *via* nonlinear lateral inhibition" *Biol. Cybernetics* 51, pp. 285–291.

Pinter, R.B. (1987) "Visual system neural networks: Feedback and feedforward lateral inhibition" *Systems and Control Encyclopedia* (ed.M.G. Singh) Oxford: Pergamon Press. pp. 5060–5065.

Pinter, R.B., Osorio, D., and Srinivasan, M.V., (in preperation) "Shift of edge preference to scototaxis depends on mean luminance and is predicted by a matched filter hypothesis in fly lamina cells"

Rall, W. (1977). "Core conductor theory and cable properties of neurons" in *Handbook of Physiology: The Nervous System* vol. I, part I, Ed. E.R. Kandel pp. 39–97. Bethesda, MD: American Physiological Society.
